# Discriminating deformable shape classes

**S. Ruiz-Correa[†], L. G. Shapiro[†], M. Meilă[‡] and G. Berson[£]**
†Department of Electrical Engineering
‡Department of Statistics
£Division of Medical Genetics, School of Medicine
University of Washington, Seattle, WA 98105

## Abstract

We present and empirically test a novel approach for categorizing 3-D free form object shapes represented by *range data* . In contrast to traditional surface-signature based systems that use alignment to match specific objects, we adapted the newly introduced *symbolic-signature representation* to classify *deformable shapes* [10]. Our approach constructs an abstract description of shape classes using an ensemble of classifiers that learn object class parts and their corresponding geometrical relationships from a set of numeric and symbolic descriptors. We used our classification engine in a series of large scale discrimination experiments on two well-defined classes that share many common distinctive features. The experimental results suggest that our method outperforms traditional numeric signature-based methodologies. [1]

## 1 Introduction

Categorizing objects from their shape is an unsolved problem in computer vision that entails the ability of a computer system to represent and generalize shape information on the basis of a finite amount of prior data. For automatic categorization to be of practical value, a number of important issues must be addressed. As pointed out in [10], how to construct a quantitative description of shape that accounts for the complexities in the categorization process is currently unknown. From a practical prospective, human perception, knowledge, and judgment are used to elaborate qualitative definitions of a class and to make distinctions among different classes. Nevertheless, categorization in humans is a standing problem in Neurosciences and Psychology, and no one is certain what information is utilized and what kind of processing takes place when constructing object categories [8]. Consequently, the task of classifying object shapes is often cast in the framework of supervised learning.

Most 3-D object recognition research in computer vision has heavily used the *alignment-verification* methodology [11] for recognizing and locating specific objects in the context of industrial machine vision. The number of successful approaches is rather diverse and spans many different axes . However, only a handful of studies have addressed the problem of categorizing shapes classes containing a significant amount of shape variation and missing information frequently found in real range scenes. Recently, Osada *et al.* [9] developed a shape representation to match similar objects. The so-called shape distribution encodes the shape information of a complete 3-D object as a probability distribution sampled from a shape function. Discrimination between classes is attempted by comparing a deterministic similarity measure based on a $L_p$ norm. Funkhouser *et al.* [1] extended the work on shape distribution by developing a representation of shape for object retrieval.

The representation is based on a spherical harmonics expansion of the points of a polygonal surface mesh rasterized into a voxel grid. Query objects are matched to the database using a nearest neighbor classifier. In [7], Martin *et al.* developed a physical model for studying neuropathological shape deformations using Principal Component Analysis and a Gaussian quadratic classifier. Golland [2] introduced the *discriminative direction* for kernel classifiers for quantifying morphological differences between classes of anatomical structures. The method utilizes the distance-transform representation to characterize shape, but it is not directly applicable to range data due to the dependence of the representation on the global structure of the objects. In [10], we developed a *shape novelty detector* for recognizing classes of 3-D object shapes in cluttered scenes. The detector learns the components of a shapes class and their corresponding geometric configuration from a set of *surface signatures* embedded in a Hilbert space. The *numeric signatures* encode characteristic surface features of the components, while the *symbolic signatures* describe their corresponding spatial arrangement.

The encouraging results obtained with our novelty detector motivated us to take a step further and extend our algorithm to accommodate classification by developing a *3-D shape classifier* to be described in the next section. The basic idea is to generalize existing surface representations that have proved effective in recognizing specific 3-D objects to the problem of object classes by using a "symbolic" representation that is resistant to deformation as opposed to a numeric representation that is tied to a specific shape. We were also motivated by applications in medical diagnosis and human interface design where 3-D shape information plays a significant role. Detecting congenital abnormalities from craniofacial features [3], identifying cancerous cells using microscopic tomography, and discriminating 3-D facial gestures are some of the driving applications.

The paper is organized as follows. Section 2 describes our proposed method. Section 3 is devoted to the experimental results. Section 4 discusses relevant aspects of our work and concludes the paper.

## 2 Our Approach

We develop our *shape classifier* in this section. For the sake of clarity we concentrate on the simplest architecture capable of performing binary classification. Nevertheless, the approach admits a straightforward extension to a multi-class setting. The basic architecture consists of a cascade of two *classification modules*. Both modules have the same structure (a bank of novelty detectors and a multi-class classifier) but operate on different input spaces. The first module processes numeric surface signatures and the second, symbolic ones. These shape descriptors characterize our classes at two different levels of abstraction.

### 2.1 Surface signatures

The surface signatures developed by Johnson and Hebert [5] are used to encode surface shape of free form objects. In contrast to the shape distributions and harmonic descriptors, their spatial scale can be enlarged to take into account local and non-local effects, which makes them robust against the clutter and occlusion generally present in range data. Experimental evidence has shown that the spin image and some of its variants are the preferred choice for encoding surface shape whenever the normal vectors of the surfaces of the objects can be accurately estimated [11]. The symbolic signatures developed in [10] are used at the next level to describe the spatial configuration of labeled surface regions.

**Numeric surface signatures.** A spin-image [5] is a two-dimensional histogram computed at an oriented point $P$ of the surface mesh of an object (see Figure 1). The histogram accumulates the coordinates $\alpha$ and $\beta$ of a set of *contributing points $Q$* on the mesh. Contributing points are those that are within a specified distance of $P$ and for which the surface normal forms an angle of less than the specified size with the surface normal $N$ of $P$. This angle is called the *support angle*. As shown in Figure 1, the coordinate $\alpha$ is the distance from $P$ to

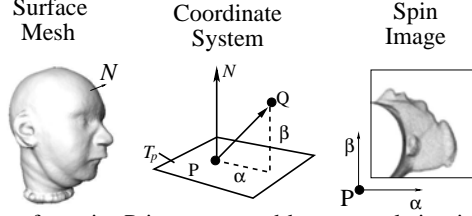

Figure 1: The spin image for point P is constructed by accumulating in a 2-D histogram the co-ordinates $\alpha$ and $\beta$ of a set of contributing points (such as Q) on the mesh representing the object.

the projection of $Q$ onto the tangent plane $T_P$ at point $P$; $\beta$ is the distance from $Q$ to this plane. We use spin images as the numeric signatures in this work.

**Symbolic surface signatures** Symbolic surface signatures (Fig. 2) are somewhat related to numeric surface signatures in that they also start with a point $P$ on the surface mesh and consider a set of contributing points $Q$, which are still defined in terms of the distance from $P$ and support angle. The main difference is that they are derived from a labeled surface mesh (shown in Figure 2a); each vertex of the mesh has an associated symbolic label referencing a surface region or *component* in which it lies. The components are constructed using a *region growing* algorithm to be described in Section 2.2. For symbolic surface signature construction, the vector $\overline{PQ}$ in Figure 2b is projected to the tangent plane at $P$ where a set of orthogonal axes $\gamma$ and $\delta$ have been defined. The direction of the $\delta - \gamma$ axes is arbitrarily defined since no curvature information was used to specify preferred directions. This ambiguity is resolved by the methods described in Section 2.2. The discretized version of the $\gamma$ and $\delta$ coordinates of $\overline{PQ}$ are used to index a 2D array, and the indexed position of the array is set to the component label of $Q$. Note thst it is possible that multiple points $Q$ that have different labels project into the same bin. In this case, the label that appeared most frequently is aasigned to the bin. The resultant array is the symbolic surface signature at point $P$. Note that the signature captures the relationships among the labeled regions on the mesh. The signature is shown as a labeled color image in Figure 2c.

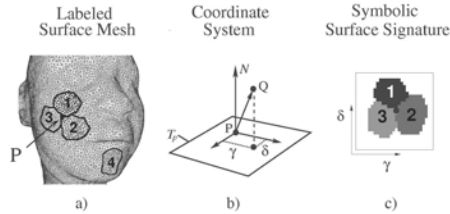

Figure 2: The symbolic surface signature for point P on a labeled surface mesh model of a human head. The signature is represented as a labeled color image for illustration purposes.

## 2.2 Classifying shape classes

We consider the classification task for which we are given a set of $l$ surface meshes $\mathbf{C} = \{C_1, \cdots, C_l\}$ representing two classes of object shapes. Each surface mesh is labeled by $y \in \{\pm 1\}$. The problem is to use the given meshes and the labels to construct an algorithm that predicts the label $y$ of a new surface mesh $C$. We let $\mathcal{C}_{+1}$ ($\mathcal{C}_{-1}$) denote the shape class labeled with $y = +1$ ($y = -1$, respectively). We start by assuming that the correspondences between all the points of the instances for each class $\mathbf{C}_y$ are known. This can be achieved by using a morphable surface models technique such as the one described in [10].

**Finding shape class components**

Before shape class learning can take place, the salient feature components associated with $\mathcal{C}_{+1}$ and $\mathcal{C}_{-1}$ must be specified . Each component of a class is identified by a particular

region located on the surface of the class members. For each class $\mathcal{C}_{+1}$ and $\mathcal{C}_{-1}$ the components are constructed one at a time using a region growing algorithm. This algorithm iteratively constructs a classification function (novelty detector), which captures regions in the space of numeric signatures $\mathcal{S}$ that approximately correspond to the support of an assumed probability distribution function $\mathcal{F}_\mathcal{S}$ associated with the class component under consideration. In this context, a shape class component is defined as the set of all mesh points of the surface meshes in a shape class whose numeric signatures lie inside of the support region estimated by the classification function. The region growing algorithm proceeds as follows.

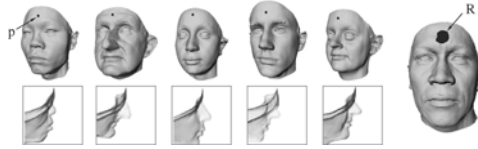

Figure 3: The component $R$ was grown around the critical point $p$ using the algorithm described in the text. Six typical models of the training set are shown. The numeric signatures for the critical point $p$ of five of the models are also shown. Their image width is 70 pixels and its region of influence covers about three quarters of the surface mesh models .

**Step I (Region Growing) .** The input of this phase is a set of surface meshes that are samples of an object class $\mathcal{C}_y$.

1. Select a set of critical points on a training object for class $\mathcal{C}_y$. Let $m_y$ be the number of critical points per object. The number $m_y$ and the locations of the critical points are chosen by hand at this time. Note that the critical points chosen for class $\mathcal{C}_+$ can differ from the critical points chosen for class $\mathcal{C}_-$.

2. Use known correspondences to find the corresponding critical points on all training instances in $\mathbf{C}$ belonging to $\mathcal{C}_y$ .

3. For each critical point $p$ of a class $\mathcal{C}_y$, compute the numeric signatures at the corresponding points of every training instance of $\mathbf{C}_y$; this set of signatures is the training set $T_{p,y}$ for critical point $p$ of class $\mathcal{C}_y$.

4. For each critical point $p$ of class $\mathbf{C}_y$, train a *component detector* (implemented as a $\nu$-SVM novelty detector [12]) to learn a component about $p$, using the training set $T_{p,y}$. The component detector will actually grow a region about $p$ using the shape information of the numeric signatures in the training sample. The regions are grown for each critical point individually using the following growing phase. Let $p$ be one of the $m$ critical points. The performance of the component detector for point $p$ can be quantified by calculating a bound on the expected probability of error $E$ on the target set as $E = \#SV_p/|\mathbf{C}_y|$, where $\#SV_p$ is the number of support vectors in the component detector for $p$, and $|\mathbf{C}_y|$ the number of elements with label $y$ in $\mathbf{C}$. Using the classifier for point $p$, perform an iterative component growing operation to expand the component about $p$. Initially, the component consists only of point $p$. An iteration of the procedure consists of the following steps. 1) Select a point that is an immediate neighbor of one of the points in the component and is not yet in the component. 2) Retrain the classifier with the current component plus the new point. 3) Compute the error $E'$ for this classifier. 4) If the new error $E'$ is lower than the previous error $E$, add the new point to the component and set $E = E'$. 5) This continues until no more neighbors can be added to the component. This region growing approach is related to the one used by Heisele *et al.* [4] for categorizing objects in 2-D images. Figure 3 shows an example of a component grown by this technique about critical point $p$ on a training set of 200 human faces from the University of South Florida database.

At the end of step I, there are $m_y$ component detectors, each of which can identify the component of a particular critical point of the object shape class $\mathbf{C}_y$. That is, when applied to a surface mesh, each component detector will determines which vertices it thinks belong to its learned component (*positive surface points*), and which vertices do not.

**Step II.** The input of this step is the training set of numeric signatures and their corresponding labels for each of the $m = m_{+1} + m_{-1}$ components. The labels are determined by the step-I component detectors previously applied to $\mathcal{C}_{+1}$ and $\mathcal{C}_{-1}$. The output is a *component classifier* (multi-class $\nu$-SVM) that, when given a *positive surface point* of a surface mesh previously processed with the bank of component detectors, will determine the particular component of the $m$ components to which this point belongs.

### Learning spatial relationships

The ensemble of component detectors and the component classifier described above define our *classification module* mentioned at the beginning of the section. A central feature of this module is that it can be used for learning the spatial configuration of the labeled components just by providing as input the set $\mathbf{C}$ of training surface meshes with each vertex labeled with the label of its component or zero if it does not belong to a component. The algorithm proceeds in the same fashion as described above except that the classifiers operate on the symbolic surface signatures of the labeled mesh. The signatures are embedded in a Hilbert space by means of a Mercer kernel that is constructed as follows. Let $A$ and $B$ be two square matrices of dimension $N$ storing arbitrary labels. Let $A * B$ denote a binary square matrix whose elements are defined as $[A * B]_{ij} = \mathrm{match}\left([A]_{ij}, [B]_{ij}\right)$, where $\mathrm{match}(a, b) = 1$ if $a = b$, and 0 otherwise. The symmetric mapping $< A, B > = (1/N^2) \sum_{ij} [A * B]_{ij}$, whose range is the interval $[0, 1]$, can be interpreted as the cosine of angle $\theta_{AB}$ between two unit vectors on the unit sphere lying within a single quadrant. The angle $\theta_{AB}$ is the geodesic distance between them. Our kernel function is defined as $k(A, B) = \exp(-\theta_{AB}^2 / \sigma^2)$.

Since symbolic surface signatures are defined up to a rotation, we use the virtual SV method for training all the classifiers involved. The method consists of training a component detector on the signatures to calculate the support vectors. Once the support vectors are obtained, new virtual support vectors are extracted from the labeled surface mesh in order to include the desired invariance; that is, a number $r$ of rotated versions of each support vector is generated by rotating the $\delta - \gamma$ coordinate system used to construct each symbolic signature (see Fig. 2). Finally, the novelty detector used by the algorithm is trained with the enlarged data set consisting of the original training data and the set of virtual support vectors.

The worse case complexity of the *classification module* is $O(nc^2s)$, where $n$ is the number of vertices of the input mesh, $s$ is the size of the input signatures (either numeric or symbolic) and $c$ is the number of novelty detectors. In the classification experiments to be described below, typical values for $n$, $s$ and $c$ are $10,000$, $2,500$ and $8$, respectively.

### A classification example

An architecture capable of discriminating two shape classes consists of a cascade of two classification modules. The first module identifies the components of each shape class, while the second verifies the geometric consistency (spatial relationships) of the components. Figure 4 illustrates the classification procedure on two sample surface meshes from a test set of 200 human heads. The first mesh (Figure 4 a) belongs to the class of healthy individuals, while the second (Figure 4 e) belongs to the class of individuals with a congenital syndrome that produces a pathological craniofacial deformation. The input classification module was trained with a set of 400 surface meshes and 4 critical points per class to recognize the eight components shown in Figure 4 b and f. The first four components are associated with healthy heads and the rest with the malformed ones. Each of the test surface meshes was individually processed as follows. Given an input surface mesh to the first classification module, the classifier ensemble (component detectors and components classifier) is applied to the numeric surface signatures of its points (Figure 4 a and e). A connected components algorithm is then applied to the result and components of size below a threshold (10 mesh points) are discarded. After this process the resulting labeled mesh is fed to the second classification module that was trained with 400 labeled meshes and two

critical points to recognize two new components. The first component was grown around the point P in Figure 4 a. The second component was grown around point Q in Figure 4 e. The symbolic signatures inside the region around P encode the geometric configuration of three of the four components learned by the first module (healthy heads), while the symbolic signatures around Q encode the geometric configuration of three of the remaining four components (malformed heads), Figure 4 b and f . Consequently, the points of the output mesh of the second module will be set to "+1" if they belong to learned symbolic signatures associated with the healthy heads (Figure 4 c) , and "-1" otherwise (Figure 4 g). Finally, the filtering algorithms described above are applied to the output mesh. Figure 4 c (g) shows the region found by our algorithm that corresponds to the shape class model of normal (respectively abnormal) head.

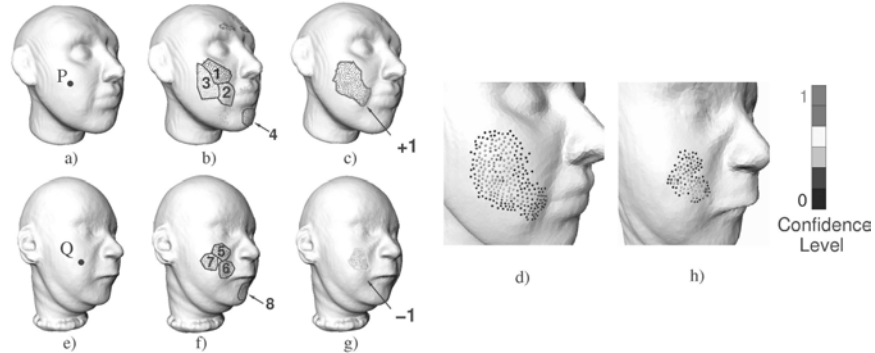

Figure 4: Binary classification example. a) and e) Mesh models of normal and abnormal heads, respectively. b) and f) Output of the first classification module. Components 1-4 are associated with healthy individuals while components 5-8, with unhealthy ones. Labeled points outside the bounded regions correspond to false positives. c) and g) Output of the second classification module. d) and h) Normalized classifier margin of the components associated with the second classification module. Red points represent high confidence values while blue points represent low values.

## 3 Experiments

We used our classifier in a series of discrimination tasks with deformable 3-D human heads and faces. All data sets were split into training and testing samples. For classification with human heads the data consisted of 600 surface mesh models (400 training samples and 200 testing samples). The models had a resolution of 1 mm ($\sim 30,000$ points) . For the faces, the data sets consisted of 300 surface meshes (200 training samples and 100 testing samples). The corresponding mesh resolution was set to about 0.8 mm ($\sim 70,000$ points). All the surface models considered here were obtained from range data scanners and all the deformable models were constructed using the methods described in [10].

We tested the stability in the formation of shape class components using the faces data set. This set contains a significant amount of shape variability. It includes models of real subjects of different gender, race, age (young and mature adults) and facial gesture (smiling vs. neutral). Typical samples are shown in Figure 3. The first module of our classifier must generate stable components to allow the second module to discriminate their corresponding geometric configurations. We trained the first classification module with a set of 200 faces using critical points arbitrarily located on the cheek, chin, forehead and philtrum of the surface models. The trained module was then applied to the testing faces to identify the corresponding components. The component associated with the forehead was correctly identified in 86% of the testing samples. This rate is reasonably high considering the amount of shape variability in the data set (Fig. 3). The percentage of identified components associated with the cheek, chin and philtrum were 86%, 89% and 82%, respectively.

We performed classification of normal versus abnormal human heads, a task that often

occurs in medical settings. The abnormalities considered are related to two genetic syndromes that can produce severe craniofacial deformities [2]. Our goal was to evaluate the performance of our classifier in discriminating examples with two well-defined where a very fine distinction exists. In our setup, the classes share many common features. This makes the classification difficult even for a trained physician. In Task I, the classifier attempted to discriminate between test samples that were 100% normal or 100% affected by each of the two model syndromes (Tasks I A and B). Task II was similar, except that the classifier was presented with examples with varying degrees of abnormality. The surface meshes of each of these examples were convex combinations of normal and abnormal heads. The degree of collusion between the resulting classes made the discrimination process more difficult. Our rationale was to drive a realistic task to its limit in order to evaluate the discrimination capabilities of the classifier. High discrimination power could be useful to quantitatively evaluate cases that are otherwise difficult to diagnose, even by human standards. The results of the experiments are summarized in Table 1. Our shape classifier was able to discriminate with high accuracy between normal and abnormal models. It was also able to discriminate classes that share a significant amount of common shape features ( see II-B$^*$ in Table 1).

We compared the performance of our approach with a signature-based method [11] that uses alignment for matching objects and is robust to scene clutter and occlusion. As we expected, a pilot study showed that the signature-based method performs poorly in tasks I A and B with an average classification rate close to 43%. The methods cited in the introduction were not considered for direct comparison, because they use global shape representations that were designed for classifying complete 3-D models. Our approach using symbolic signatures can operate on single-view data sets containing partial model information, as shown by the experimental results performed on several shape classes [10].

| I-A (100% normal - 0% abnormal) | 98 | II-B (50% normal - 50% abnormal) | 97 |
|---|---|---|---|
| I-B (100% normal - 0% abnormal) | 100 | II-B $^*$ (25% normal - 75% abnormal) | 92 |
| II-B (65% normal - 35% abnormal) | 98 | II-B (15% normal - 85% abnormal) | 48 |

Table 1: Classification accuracy rate (%) for discrimination between above test samples versus 100% abnormal test samples.

## 4 Discussion and Conclusion

We presented a supervised approach to classification of 3-D shapes represented by range data that learns class components and their geometrical relationships from surface descriptors. We performed preliminary classification experiments on models of human heads (normal vs. abnormal) and studied the stability in the formation of class components using a collection of real face models containing a large amount of shape variability. We obtained promising results. The classification rates were high and the algorithm was able to grow consistent class components despite the variance.

We want to stress which parts of our approach are essential as described and which are modifiable. The numeric and symbolic shape descriptors considered here are important. They are locally defined but they convey a certain amount of global information. For example, the spin image defined on the forehead (point P) in Figure 3 encodes information about the shape of most of the face (including the chin). As the image width increases, the spin image becomes more descriptive. Spin images and some variants [11] are reliable for encoding surface shape in the present context. Other descriptors such as curvature-based or harmonic signatures are not descriptive enough or lack robustness to scene clutter and occlusion. In the classification experiments described above, we did not perform any kind of feature selection for choosing the critical points. Nevertheless, the shape descriptors cap-

tured enough global information to allow a classifier to discriminate between the distinctive features of normal and abnormal heads.

The structure of the classification module (bank of novelty detectors and multi-class classifier) is important. The experimental results showed us that the output of the novelty detectors is not always reliable and the multi-class classifier becomes critical for constructing stable and consistent class components. In the context of our medical application, the performance of our novelty detectors can be improved by incorporating prior information into the classification scheme. Maximum entropy classifiers or an extension of the Bayes point machines to the one class setting are being investigated as possible alternatives. The region-growing algorithm for finding class components is not critical. The essential point consists of generating groups of neighboring surface points whose shape descriptors are similar but distinctive enough from the signatures of other components.

There are several issues to investigate. 1) Our method is able to model shape classes containing significant shape variance and can absorb about 20% of scale changes. A multi-resolution approach could be used for applications that require full scale invariance. 2) We used large range data sets for training our classifier. However, larger sets are required in order to capture the shape variability of the abnormal craniofacial features due to race, age and gender. We are currently collecting data from various medical sources to create a database for implementing and testing a semi-automated diagnosis system. The data includes 3-D models constructed from range data and CT scans. The usability of the system will be evaluated by a panel of expert geneticists.

## Footnotes

[1]This research is based upon work supported by NSF Grant No. IIS-0097329 and NIH Grant No. P20LM007714. Any opinions, findings and conclusions or recomendations expressed in this material and those of the autors do not necessarily reflects the views of NSF o NIH.

[2]Test samples were obtained from models with craniofacial features based upon either the Greig cephalopolysyndactyly (A) or the trisomy 9 mosaic (B) syndromes [6].

## References

[1] T. Funkhouser, P. Min, M. Kazhdan, J. Chen, A. Halderman, D. Dobkin, and D. Jacobs "A Search Engine for 3D Models," *ACM Transactions on Graphics*, 22(1), pp. 83-105, January 2003.

[2] P. Golland "Discriminative Direction for Kernel Classifiers," *In: Advances in Neural Information Processing Systems*, 13, Vancouver, Canada, 745-752, 2001.

[3] P. Hammond, T. J. Hunton, M. A. Patton, and J. E. Allanson. "Delineation and Visualization of Congenital Abnormality using 3-D Facial Images," *In:Intelligent Data Analysis in Medicine and Pharmacology*, MEDINFO, 2001, London.

[4] B. Heisele, T. Serre, M. Pontil, T. Vetter and T. Poggio. "Categorization by Learning and Combining Object Parts," *In: Advances in Neural Information Processing Systems*, 14, Vancouver, Canada, Vol. 2, 1239-1245, 2002.

[5] A. E. Johnson and M. Hebert, "Using Spin Images for Efficient Object Recognition in Cluttered 3D scenes," *IEEE Trans. Pattern Analysis and Machine Intelligence*, 21(5), pp. 433-449, 1999.

[6] K. L. Jones, *Smith's Recognizable Patterns of Human Malformation*, 5th Ed. W.B. Saunders Company, 1999.

[7] J. Martin, A. Pentland, S. Sclaroff, and R. Kikinis, "Characterization of Neurophatological Shape Deformations," *IEEE Transactions on Pattern Analysis and Machine Intelligence,*, Vol. 2, No. 2, 1998.

[8] D. L. Medin, C. M. Aguilar, *Categorization*. In R.A. Wilson and F. C. Keil (Eds.). The MIT Encyclopedia of the Cognitive Sciences, Cambridge, MA, 1999.

[9] R. Osada, T. Funkhouser, B. Chazelle, and D. Dobkin, "Matching 3-D models with shape distributions," *Shape Modeling International*, 2001, pp. 154-166.

[10] S. Ruiz-Correa, L. G. Shapiro, and M. Meilă. "A New Paradigm for Recognizing 3-D Object Shapes from Range Data," *Proceedings of the IEEE Computer Society International Conference on Computer Vision 2003*, Vol.2, pp. 1126-1133.

[11] S. Ruiz-Correa, L. G. Shapiro, and M. Meilă, "A New Signature-based Method for Efficient 3-D Object Recognition," *Proceedings of the IEEE Conference on Computer Vision and Pattern Recognition 2001*, Vol. 1, pp. 769 -776.

[12] B. Scholköpf and A. J. Smola, *Learning with Kernels,* The MIT Press, Cambridge, MA, 2002.
